# Diffusion Approximations for the Constant Learning Rate Backpropagation Algorithm and Resistence to Local Minima

**William Finnoff**
Siemens AG, Corporate Research and Development
Otto-Hahn-Ring 6
8000 Munich 83, Fed. Rep. Germany

## Abstract

In this paper we discuss the asymptotic properties of the most commonly used variant of the backpropagation algorithm in which network weights are trained by means of a local gradient descent on examples drawn randomly from a fixed training set, and the learning rate $\eta$ of the gradient updates is held constant (simple backpropagation). Using stochastic approximation results, we show that for $\eta \to 0$ this training process approaches a batch training and provide results on the rate of convergence. Further, we show that for small $\eta$ one can approximate simple back propagation by the sum of a batch training process and a Gaussian diffusion which is the unique solution to a linear stochastic differential equation. Using this approximation we indicate the reasons why simple backpropagation is less likely to get stuck in local minima than the batch training process and demonstrate this empirically on a number of examples.

## 1   INTRODUCTION

The original (simple) backpropagation algorithm, incorporating pattern for pattern learning and a constant learning rate $\eta \in (0, \infty)$, remains in spite of many real (and

imagined) deficiencies the most widely used network training algorithm, and a vast body of literature documents its general applicability and robustness. In this paper we will draw on the highly developed literature of stochastic approximation theory to demonstrate several asymptotic properties of simple backpropagation. The close relationship between backpropagation and stochastic approximation methods has been long recognized, and various properties of the algorithm for the case of decreasing learning rate $\eta_{n+1} < \eta_n$, $n \in \mathbf{N}$ were shown for example by White [W,89a], [W,89b] and Darken and Moody [D,M,91]. Hornik and Kuan [H,K,91] used comparable results for the algorithm with constant learning rate to derive weak convergence results.

In the first part of this paper we will show that simple backpropagation has the same asymptotic dynamics as batch training in the small learning rate limit. As such, anything that can be expected of batch training can also be expected in simple backpropagation as long as the learning rate of the algorithm is very small. In the special situation considered here (in contrast to that in [H,K,91]) we will also be able to provide a result on the speed of convergence. In the next part of the paper, Gaussian approximations for the difference between the actual training process and the limit are derived. It is shown that this difference, (properly renormalized), converges to the solution of a linear stochastic differential equation. In the final section of the paper, we combine these results to provide an approximation for the simple backpropagation training process and use this to show why simple backpropagation will be less inclined to get stuck in local minima than batch training. This ability to avoid local minima is then demonstrated empirically on several examples.

## 2    NOTATION

Define the parametric version of a single hidden layer network activation function with $h$ inputs, $m$ outputs and $q$ hidden units

$$f : \mathbf{R}^d \times \mathbf{R}^h \to \mathbf{R}^m, (\theta, x) \to (f^1(\theta, x), ..., f^m(\theta, x)),$$

by setting for $x \in \mathbf{R}^h$, $\bar{x} = (x_1, ..., x_h, 1)$, $\theta = (\gamma., \beta.)$ and $u = 1, ..., m$,

$$f^u(\theta, x) = f^u((\gamma., \beta.), x) = \phi \left( \sum_{j=1}^{q} \gamma_j^u \phi(\beta_j \bar{x}^T) + \gamma_{q+1}^u \right),$$

where $\bar{x}^T$ denotes the transpose of $\bar{x}$ and $d = m(q + 1) + q(h + 1)$ denotes the number of weights in the network. Let $((Y_k, X_k))_{k=1,...,T}$ be a set of training examples, consisting of targets $(Y_k)_{k=1,...,T}$ and inputs $(X_k)_{k=1,...,T}$. We then define the parametric error function

$$U(y, x, \theta) = \|y - f(\theta, x)\|^2, .$$

and for every $\theta$, the cummulative gradient

$$h(\theta) = -\frac{1}{T} \sum_{k=1}^{T} \frac{\partial U}{\partial \theta}(Y_k, X_k, \theta).$$

# 3   APPROXIMATION WITH THE ODE

We will be considering the asymptotic properties of network training processes induced by the starting value $\theta_0$, the gradient (or direction) function $-\frac{\partial U}{\partial \theta}$ the learning rate $\eta$ and an infinite training sequence $(y_n, x_n)_{n \in \mathbf{N}}$, where each $(y_n, x_n)$ example is drawn at random from the set $\{(Y_1, X_1), ..., (Y_T, X_T)\}$. One defines the discrete parameter process $\theta = \theta^\eta = (\theta_n^\eta)_{n \in \mathbf{Z}_+}$ of weight updates by setting

$$\theta_n^\eta = \begin{cases} \theta_0 & \text{for } n = 0 \\ \theta_{n-1}^\eta - \eta \frac{\partial U}{\partial \theta}(y_n, x_n, \theta_{n-1}^\eta) & \text{for } n \in \mathbf{N} \end{cases}$$

and the corresponding continuous parameter process $(\theta^\eta(t))_{t \in [0,\infty)}$, by setting

$$\theta_n^\eta(t) = \theta_{n-1}^\eta - (t - (n-1)\eta)\frac{\partial U}{\partial \theta}(y_n, x_n, \theta_{n-1}^\eta)$$

for $t \in [(n-1)\eta, n\eta)$,  $n \in \mathbf{N}$. The first question that we will investigate is that of the 'small learning rate limit' of the continuous parameter process $\theta^\eta$, i.e. the limiting properties of the family $\theta^\eta$ for $\eta \to 0$. We show that the family of (stochastic) processes $(\theta^\eta)_{\eta > 0}$ converges with probability one to a limit process $\overline{\theta}$, where $\overline{\theta}$ denotes the solution to the cummulative gradient equation,

$$\overline{\theta}(t) = \theta_0 + \int_0^t h(\overline{\theta}(s))ds.$$

Here, for $\theta_0 = a = constant$, this solution is deterministic. This result corresponds to a 'law of large numbers' for the weight update process, in which the small learning rate (in the limit) averages out the stochastic fluctuations.

Central to any application of the stochastic approximation results is the derivation of local Lipschitz and linear growth bounds for $\frac{\partial U}{\partial \theta}$ and $h$. That is the subject of the following,

**Lemma(3.1)** i) There exists a constant $K > 0$ so that

$$\sup_{(y,x)} \left\| \frac{\partial U}{\partial \theta}(y, x, \theta) \right\| \le K(1 + \|\theta\|)$$

and

$$\|h(\theta)\| \le K(1 + \|\theta\|).$$

ii) For every $G > 0$ there exists a constant $L_G$ so that for any $\theta, \widetilde{\theta} \in [-G, G]^d$,

$$\sup_{(y,x)} \left\| \frac{\partial U}{\partial \theta}(y,x,\theta) - \frac{\partial U}{\partial \theta}(y,x,\widetilde{\theta}) \right\| \leq L_G \|\theta - \widetilde{\theta}\|$$

and

$$\|h(\theta) - h(\widetilde{\theta})\| \leq L_G \|\theta - \widetilde{\theta}\|.$$

Proof: The calculations on which this result are based are tedious but straightforward, making repeated use of the fact that products and sums of locally Lipschitz continuous functions are themselves locally Lipschitz continuous. It is even possible to provide explicit values for the constants given above.

●

Denoting with **P** (resp. **E**) the probability (resp. mathematical expectation) of the processes defined above, we can present the results on the probability of deviations of the process $\theta$ from the limit $\overline{\theta}$.

**Theorem(3.2)** Let $r, \delta \in (0, \infty)$. Then there exists a constant $B_r$ (which doesn't depend on $\eta$) so that

i) $\mathbf{E}\left(\sup_{s \leq r} \|\theta^\eta(s) - \overline{\theta}(s)\|^2\right) \leq B_r \eta$.

ii) $\mathbf{P}\left(\sup_{s \leq r} \|\theta(s) - \overline{\theta}(s)\| > \delta\right) \leq \frac{1}{\delta^2} B_r \eta$.

Proof: The first part of the proof requires that one finds bounds for $\theta^\eta(t)$ and $\overline{\theta}(t)$ for $t \in [0, r]$. This is accomplished using the results of Lemma(3.1) and Gronwall's Lemma. This places $\eta$ independent bounds on $B_r$. The remainder of the proof uses Theorem(9), §1.5, Part II of [Ben,Met,Pri,87]. The required conditions (A1), (A2) follow directly from our hypotheses, and (A3), (A4) from Lemma(3.1). Due to the boundedness of the variables $(y_n, x_n)_{n \in \mathbf{N}}$ and $\theta_0$, condition (A5) is trivially fulfilled.

●

It should be noted that the constant $B_r$ is usually dependent on $r$ and may indeed increase exponentially (in $r$) unless it is possible to show that the training process remains in some bounded region for $t \to \infty$. This is not necessarily due exclusively to the difference between the stochastic approximation and the discrete parameter cummulative gradient process, but also to the the error between the discrete (Euler approximation) and continuous parameter versions of (3.3).

## 4    GAUSSIAN APPROXIMATIONS

In this section we will give a Gaussian approximation for the difference between the training process $\theta^\eta$ and the limit $\overline{\theta}$. Although in the limit these coincide, for $\eta > 0$ the training process fluctuates away from the limit in a stochastic fashion. The following Gaussian approximation provides an estimate for the size and nature

of these fluctuations depending on the second order statistics (variance/covariance matrix) of the weight update process. Define for any $t \in [0, \infty)$,

$$\Theta^\eta(t) = \frac{\theta^\eta(t) - \overline{\theta}(t)}{\sqrt{\eta}}.$$

Further, for $i = 1, ..., d$ we denote with $\frac{\partial U}{\partial \theta}^i(y, x, \theta)$, (resp. $h^i(\theta)$) the $i$-th coordinate vector of $\frac{\partial U}{\partial \theta}(y, x, \theta)$ (resp. $h(\theta)$). Then define for $i, j = 1, ..., d, \theta \in \mathbf{R}^d$

$$R^{ij}(\theta) = \frac{1}{T} \sum_{k=1}^{T} \left( \frac{\partial U^i}{\partial \theta}(Y_k, X_k, \theta) - h^i(\theta) \right) \left( \frac{\partial U^j}{\partial \theta}(Y_k, X_k, \theta) - h^j(\theta) \right).$$

Thus, for any $n \in \mathbf{N}$, $\theta \in \mathbf{R}^d$, $R(\theta)$ represents the covariance matrix of the random elements $\frac{\partial U}{\partial \theta}(y_n, x_n, \theta)$. We can then define for the symmetric matrix $R(\theta)$ a further $\mathbf{R}^{d \times d}$ valued matrix $R^{\frac{1}{2}}(\theta)$ with the property that $R(\theta) = R^{\frac{1}{2}}(\theta)(R^{\frac{1}{2}}(\theta))^T$.

The following result represents a central limit theorem for the training process. This permits a type of second order approximation of the fluctuations of the stochastic training process around its deterministic limit.

**Theorem(4.1):** Under the assumptions given above, the distributions of the processes $\Theta^\eta$, $\eta > 0$, converge weakly (in the sense of weak convergence of measures) for $\eta \to 0$ to a uniquely defined measure $\mathcal{L}\{\widetilde{\theta}\}$, where $\widetilde{\theta}$ denotes the solution to the following stochastic differential equation

$$\widetilde{\theta}(t) = \int_0^t \frac{\partial h}{\partial \theta}(\overline{\theta}(s))\widetilde{\theta}(s)ds + \int_0^t R^{\frac{1}{2}}(\overline{\theta}(s))dW(s),$$

where $W$ denotes a standard $d-$dimensional Brownian motion (i.e. with covariance matrix equal to the identity matrix).

Proof: The proof here uses Theorem(7), §4.4, Part II of [Ben,Met,Pri,87]. As noted in the proof of Theorem(3.2), under our hypotheses, the conditions (A1)-(A5) are fulfilled. Define for $i, j = 1, ..., d$, $(y, x) \in I^{m+h}$, $\theta \in \mathbf{R}^d$, $w^{ij}(y, x, \theta) = \rho^i(y, x, \theta)\rho^j(y, x, \theta) - h^i(\theta)h^j(\theta)$, and $\nu = \rho$. Under our hypotheses, $h$ has continuous first and second order derivatives for all $\theta \in \mathbf{R}^d$ and the function $R = (R^{ij})_{i,j=1,...,d}$ as well as $W = (w^{ij})_{i,j=1,...,d}$ fulfill the remaining requirements of (A8) as follows: (A8)i) and (A8)ii) are trivial consequence of the definition of $R$ and $W$. Finally, setting $p_3 = p_4 = 0$ and $\mu = 1$, (A8)iii) then can be derived directly from the definitions of $W$ and $R$ and Lemma(5.1)ii).

●

# 5    RESISTENCE TO LOCAL MINIMA

In this section we combine the results of the two preceding sections to provide a Gaussian approximation of simple backpropagation. Recalling the results and

notation of Theorem(3.2) and Theorem(4.1) we have for any $t \in [0, \infty)$,

$$\theta^\eta(t) = \bar{\theta}(t) + \eta^{\frac{1}{2}}\tilde{\theta}(t) + o(\eta^{\frac{1}{2}}).$$

Using this approximation we have:

-For 'very small' learning rate $\eta$, simple backpropagation and batch learning will produce essentially the same results since the stochastic portion of the process (controlled by $\eta^{\frac{1}{2}}$) will be negligible.

-Otherwise, there is a nonnegligible stochastic element in the training process which can be approximated by the Gaussian diffusion $\tilde{\theta}$.

-This diffusion term gives simple backpropagation a 'quasi-annealing' character, in which the cummulative gradient is continuously perturbed by the Gaussian term $\tilde{\theta}$, allowing it to escape local minima with small shallow basins of attraction.

It should be noted that the rest term will actually have a better convergence rate than the indicated $o(\eta^{\frac{1}{2}})$. The calculation of exact rates, though, would require a generalized version of the Berry-Esséen theorem. To our knowledge, no such results are available which would be applicable to the situation described above.

## 6   EMPIRICAL RESULTS

The imperviousness of simple backpropagation to local minima, which is part of neural network 'folklore' is documented here in four examples. A single hidden layer feedforward network with $\phi = tanh$, ten hidden units n and one output was trained with both simple backpropagation and batch training using data generated by four different models. The data consisted of pairs $(y_i, x_i)$, $i = 1, ..., T$, $T \in \mathbf{N}$ with targets $y_i \in \mathbf{R}$ and inputs $x_i = (x_i^1, ..., x_i^K) \in [-1, 1]^K$, where $y_i = g((x_i^1, ..., x_i^j)) + u_i$, for $j, K \in \mathbf{N}$. The first experiment was based on an additive structure $g$ having the following form with $j = 5$ and $K = 10$, $g((x_i^1, ..., x_i^5)) = \sum_{k=1}^5 \sin(\alpha^k x_i^k)$, $\alpha^k \in \mathbf{R}$. The second model had a product structure $g$ with $j = 3$, $K = 10$ and $g((x_i^1, ..., x_i^3)) = \prod_{k=1}^3 x_i^k$, $\alpha^k \in \mathbf{R}$. The third structure considered was constructed with $j = 5$ and $K = 10$, using sums of Radial Basis Functions (RBF's) as follows: $g((x_i^1, ..., x_i^5)) = \sum_{l=1}^8 (-1)^l \exp\left(\sum_{k=1}^5 \frac{(\alpha^{k,l} - x_i^k)^2}{2\sigma^2}\right)$. These points were chosen by independent drawings from a uniform distribution on $[-1, 1]^5$. The final experiment was conducted using data generated by a feedforward network activation function. For more details concerning the construction of the examples used here consult [F,H,Z,92].

For each model three training runs were made using the same vector of starting weights for both simple backpropagation and batch training. As can be seen, in all but one example the batch training process got stuck in a local minimum producing much worse results than those found using simple backpropagation. Due to the wide array of structures used to generate data and the number of data sets used, it would be hard to dismiss the observed phenomena as being example dependent.

## Net

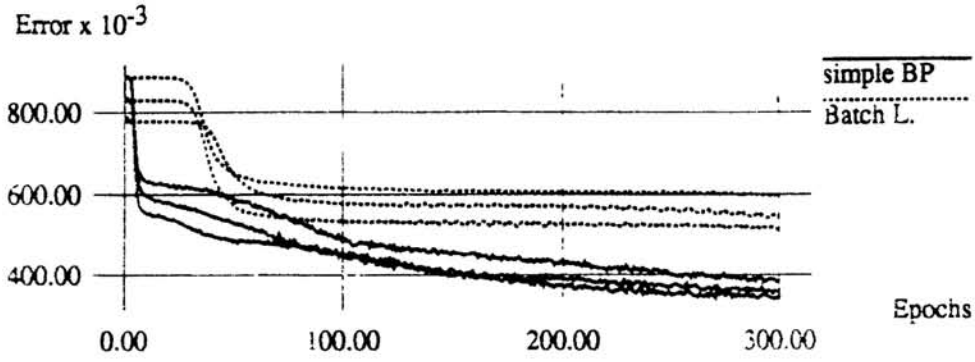

## Product Mapping

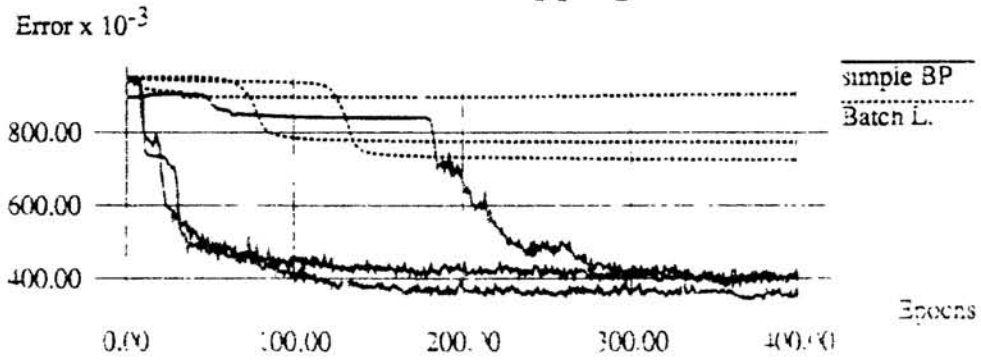

## Sums of RBF's

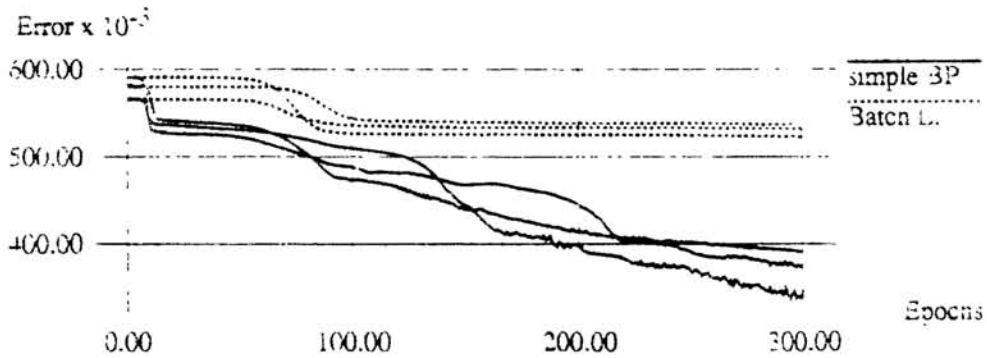

## Sums of sin's

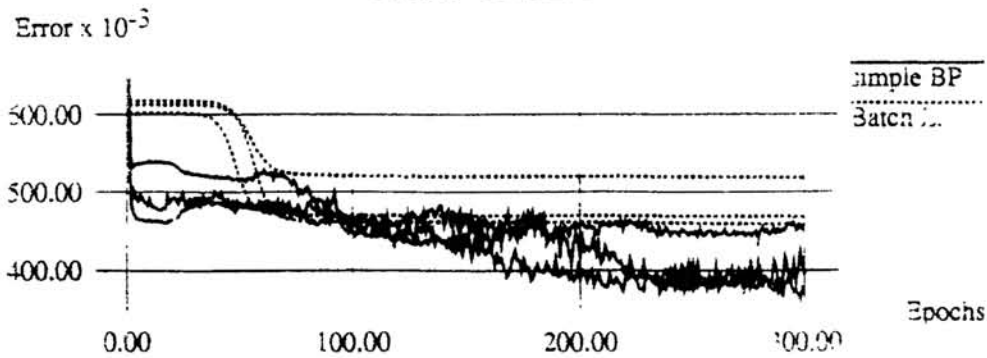

## 7    REFERENCES

[Ben,Met,Pri,87] Benveniste, A., Métivier, M., Priouret, P., *Adaptive Algorithms and Stochastic Approximations*, Springer Verlag, 1987.

[Bou,85] Bouton C., Approximation Gaussienne d'algorithmes stochastiques a dynamique Markovienne. Thesis, Paris VI, (in French), 1985.

[Da,M,91] Darken C. and Moody J., Note on learning rate schedules for stochastic optimization, in *Advances in Neural Information Processing Systems 3*, Lippmann, R. Moody, J., and Touretzky, D., ed., Morgan Kaufmann, San Mateo, 1991.

[F,H,Z,92] Improving model selection by nonconvergent methods. To appear in *Neural Networks*.

[H,K,91], Hornik, K. and Kuan, C.M., Convergence of Learning Algorithms with constant learning rates, *IEEE Trans. on Neural Networks* 2, pp. 484-489, (1991).

[Wh,89a] White, H., Some asymptotic results for learning in single hidden-layer feedforward network models, *Jour. Amer. Stat. Ass.* 84, no. 408, p. 1003-1013, 1989.

[W,89b] White, H., Learning in artificial neural networks: A statistical perspective, *Neural Computation* 1, p.425-464, 1989.
